# Model Matching and SFMD Computation

**Steve Rehfuss and Dan Hammerstrom**
Department of Computer Science and Engineering
Oregon Graduate Institute of Science and Technology
P.O.Box 91000, Portland, OR 97291-1000 USA
*stever@cse.ogi.edu, strom@asi.com*

## Abstract

In systems that process sensory data there is frequently a *model matching* stage where class hypotheses are combined to recognize a complex entity. We introduce a new model of parallelism, the *Single Function Multiple Data (SFMD)* model, appropriate to this stage. SFMD functionality can be added with small hardware expense to certain existing SIMD architectures, and as an incremental addition to the programming model. Adding SFMD to an SIMD machine will not only allow faster model matching, but also increase its flexibility as a general purpose machine and its scope in performing the initial stages of sensory processing.

## 1 INTRODUCTION

In systems that process sensory data there is frequently a post-classification stage where several independent class hypotheses are combined into the recognition of a more complex entity. Examples include matching word models with a string of observation probabilities, and matching visual object models with collections of edges or other features. Current parallel computer architectures for processing sensory data focus on the classification and pre-classification stages (Hammerstrom 1990).This is reasonable, as those stages likely have the largest potential for speedup through parallel execution. Nonetheless, the *model-matching* stage is also suitable for parallelism, as each model may be matched independently of the others.

We introduce a new style of parallelism, *Single Function Multiple Data (SFMD)*, that is suitable for the model-matching stage. The handling of interprocessor synchronization distinguishes the SFMD model from the SIMD and MIMD models: SIMD synchronizes implicitly at each instruction, SFMD synchronizes implicitly at conditional expression or loop boundaries, and MIMD synchronizes explicitly at

arbitrary inter-processor communication points. Compared to MIMD, the use of implicit synchronization makes SFMD easier to program and cheaper to implement. Compared to SIMD, the larger granularity of synchronization gives SFMD increased flexibility and power.

SFMD functionality can be added with small hardware expense to SIMD architectures already having a high degree of processor autonomy. It can be presented as an incremental addition to programmer's picture of the machine, and applied as a compiler optimization to existing code written in an SIMD version of 'C'. Adding SFMD to an SIMD machine will not only allow faster model matching, but also increase its flexibility as a general purpose machine, and increase its scope in performing the initial stages of sensory processing.

## 2  SIMD ARCHITECTURE AND PROGRAMMING

As background, we first review SIMD parallelism. In SIMD, multiple processing elements, or *PE*'s, simultaneously execute identical instruction sequences, each processing different data. The instruction stream is produced by a controller, or *sequencer*. Generally, each PE has a certain amount of *local memory*, which only it can access directly. All PEs execute a given instruction in the stream at the same time, so are synchronized at each instruction. Thus synchronization is implicit, the hardware need not support it, and the programmer need (can) not manage it. SIMD architectures differ in the functionality of their PEs. If PEs can independently address local memory at differing locations, rather than all having to access the same address at a given step, the architecture is said to have *local addressing*. If PEs can independently determine whether to execute a given instruction, rather than having this determined by the sequencer, the architecture has *local conditional execution*. Note that all PEs see the same instruction stream, yet a given PE executes only one branch of any if-then-else, and so must idle while other PEs execute the other branch. This is the cost of synchronizing at each instruction.

## 3  MODEL MATCHING

We view models as pieces of *a priori* knowledge, interrelating their *components*. Models are *matched* against some *hypothesis set* of possible features. Matching produces a *correspondence* between components of the model and elements of the hypothesis set, and also *aligns* the model and the set ("pose estimation" in vision, and "time-alignment" in speech). An essential fact is that, because models are known *a priori*, in cases where there are many models it is usually possible and profitable to construct an *index* into the set of models. Use of the index at runtime restricts the set of models that need actually be matched to a few, high-probability ones.

Model-matching is a common stage in sensory data processing. Phoneme, character and word HMMs are models, where the hypothesis set is a string of observations and the matching process is either of the usual Viterbi or trellis procedures. For phonemes and characters, the HMMs used typically all have the same graph structure, so control flow in the matching process is not model-dependent and may be encoded in the instruction stream. Word models have differing structure, and control flow is model-dependent. In vision, model-matching has been used in a variety of complicated ways (cf. (Suetens, Fua & Hanson 1992)), for example, graph models may have constraints between node attribute values, to be resolved during matching.

# 4   DATA AND KNOWLEDGE PARALLELISM

SIMD is a type of computer architecture. At the algorithm level, it corresponds to *data parallelism*. Data parallelism, applying the same procedure in parallel to multiple pieces of data, is the most common explicit parallelization technique.and is the essence of the *Single Program Multiple Data (SPMD)* programming model. On a distributed memory machine, SPMD can be stylized as "given a limited amount of (algorithmic) knowledge to be applied to a large piece of data, distribute the data and broadcast the knowledge".

In sensory processing systems, conversely, one may have a large amount of knowledge (many models) that need to be applied to a (smallish) piece of data, for example, a speech signal frame or segment, or a restricted region of an image. In this case, it makes sense to "distribute the knowledge and broadcast the data". Model-matching often works well on an SIMD architecture, e.g. for identical phoneme models. However, when matching requires differing control flow between models, an SIMD implementation can be inefficient.

Data and knowledge parallelism are asymmetrical, however, in two ways. First, all data must normally be processed, while there are usually indexing techniques that greatly restrict the number of models that actually must be matched. Second, processing an array element frequently requires information about neighboring elements; when the data is partitioned among multiple processors, this may require inter-processor communication and synchronization. Conversely, models on different processors can be matched to data in their local memories without any inter-processor communication. The latter observation leads to the SFMD model.

# 5   PROGRAMMING MODEL

We view support for SFMD as functionality to be added to an existing SIMD machine to increase its flexibility, scope, and power. As such, the SFMD programming model should be an extension of the SIMD one. Given an SIMD architecture with the local addressing and local conditional execution, SFMD programming is made available at the assembly language level by adding three constructs:

**distribute n** tells the sequencer and PEs that the next *n* instructions are to be distributed for independent execution on the PEs. We call the next *n* instructions an *SFMD block*.

**sync** tells the individual PEs to suspend execution and signal the controller (barrier synchronization). This is a no-op if not within an SFMD block.

**branch-local** one or more local branch instruction(s), including a loop construct; the branch target must lie within the enclosing SFMD block. This is a no-op if not within an SFMD block.

We further require that *code within an SFMD block contain only references to PE-local memory*; none to global (sequencer) variables, to external memory or to the local memory of another PE. It must also contain no inter-PE communication.. When the PEs are independently executing an SFMD block, we say that the system is in *SFMD mode*, and refer to normal execution as *SIMD mode*.

When programming in a data-parallel 'C'-like language for an SIMD machine, use of SFMD functionality can be an optimization performed by the compiler, completely hidden from the user. Variable type and usage analysis can determine for any given block of code whether the constraints on non-local references are met, and emit

code for SFMD execution if so. No new problems are introduced for debugging, as SFMD execution is semantically equivalent to executing on each PE sequentially, and can be executed this way during debugging.

To the programmer, SFMD ameliorates two inefficiencies of SIMD programming: (i) in conditionals, a PE need not be idle while other PEs execute the branch it didn't take, and (ii) loops and recursions may execute a processor-dependent number of times.

# 6  HARDWARE MODEL AND COST

We are interested in embedded, "delivery system" applications. Such systems must have few chips; scalability to 100's or 1000's of chips is not an issue. Parallelism is thus achieved with multiple PEs per chip. As off-chip I/O is always expensive compared to computation[1], such chips can contain only a relatively small number of processors. Thus, as feature size decreases, area will go to local memory and processor complexity, rather than more processors.

Adding SFMD functionality to an architecture whose PEs have local addressing and local conditional execution is straightforward. Here we outline an example implementation. Hardware for branch tests and decoding sequencer instructions in the instruction register (IR) already exists. Local memory is suitable for local addressing. A very simple "micro-sequencer" must be added, consisting essentially of a program counter (PC) and instruction buffer (IM), and some simple decode logic. The existing PE output path can be used for the barrier synchronization. A 1-bit path from the sequencer to each PE is added for interrupting local execution.

Execution of a `distribute` n instruction on a PE causes the next $n$ instructions to be stored sequentially in IM, starting at the current address in the PC. The $(n+1)$'st instruction is executed in SPMD mode, it is typically either a `branch-local` to start execution, or possibly a `sync` if the instructions are just being cached[2].

Almost the entire cost of providing SFMD functionality is silicon area used by the IM. The IM contains inner loop code, or model-driven conditional code, which is likely to be small. For a 256 4-byte instruction buffer on the current ASI CNAPS 1064, having 64 PEs with 4KB memory each, this is about 11% of the chip area; for a hypothetical 16 PE, 16K per PE chip of the same size, it is 3%. These numbers are large, but as feature size decreases, the incremental cost of adding SFMD functionality to an SIMD architecture quickly becomes small.

# 7  PERFORMANCE

What performance improvement may be expected by adding SFMD to SIMD? There are two basic components, improvement on branches, and improvement on nested loops, where the inner loop count varies locally.

Unnested (equiprobable) branches speed up most when the branch bodies have the same size, with a factor of 2 improvement. For nested branches of depth $d$, the factor is $2^d$, but these are probably unusual. An exception would be applying a decision tree classifier in a data-parallel way.

To examine improvement on nested loops, suppose we have a set of $N$ models (or any independent tasks) to be evaluated on an architecture with $P$ processors. On

an SFMD architecture, we partition the set into $P$ groups, assign each group to a processor, and have each processor evaluate all the models in its group. If evaluating the $j$'th model of the $i$'th group takes time $t_{ij}^{(sfmd)}$, then the total time is

$$T_{sfmd} = \max_{i=1}^{P} \sum_{j=1}^{N_i} t_{ij}^{(sfmd)} \tag{1}$$

where $N_i$ is the size of the $i$'th group, $\sum_{i=1}^{P} N_i = N$. On an SIMD architecture, we partition the set into $\lceil N/P \rceil$ groups of size $P$ and sequentially evaluate each group in parallel. Each group has a model that takes the most time to evaluate; SIMD execution forces the whole group to have this time complexity. So, evaluating a single group, $G_i$, takes time $\max_j t_{ij}^{(simd)}$, where $j$ indexes over the elements of the group, $1 \le j \le P$. The total time for SIMD execution is then

$$T_{simd} = \sum_{i=1}^{\lceil N/P \rceil} \max_{j=1}^{P} t_{ij}^{(simd)} \tag{2}$$

Ignoring data-dependent branching and taking $t_{ij}^{(simd)} = t_{ij}^{(sfmd)} \doteq t_{ij}$, we see that optimal $(i,j)$-indexing of the $N$ models for either case is a bin packing problem . As such, $(i,j)$-indexing will be heuristic, and we examine $T_{simd}/T_{sfmd}$ by simulation. It should be clear that the expected improvement due to SFMD cannot be large unless the outer loop count is large. So, for model matching, improvement on nested loops is likely not an important factor, as usually only a few models are matched at once.

To examine the possible magnitude of the effect in general, we look instead at multiplication of an input vector by a large sparse matrix. Rows are partitioned among the PEs, and each PE computes all the row-vector inner products for its set of rows[3]. $T_{sfmd}$ is given by equation (1), with $\{t_{ij} | 1 \le j \le N_i\}$ the set of all rows for processor $i$. $T_{simd}$ is given by equation (2), with $\{t_{ij} | 1 \le j \le P\}$ the set of rows executed by all processors at time $i$. Here $t_{ij}$ is the time to perform a row-vector inner product.

Under a variety of choices of matrix size ($256 \times 256$ to $2048 \times 2048$), number of processors (16,32,64), distribution of elements (uniform, clustered around the diagonal), and sparsity (fraction of nonzero elements from 0.001 to 0.4) we get that the ratio $T_{simd}/T_{sfmd}$ decreases from around 2.2-2.7 for sparsities near 0.001, to 1.2 for sparsities near 0.06, and to 1.1 or less for more dense matrices (Figure 1). The effect is thus not dramatic.

As an example of the potential utility of SFMD functionality for model matching, we consider *interpretation tree search (ITS)*, a technique used in vision[4]. ITS is a technique for establishing a correspondence between image and model features. It consists essentially of depth-first search (*DFS*), where a node on level $d$ of the tree corresponds to a pairing of image features with the first $d$ model features. The search is limited by a variety of unary and binary geometric constraints on the allowed pairings. Search complexity implies small models are matched to small

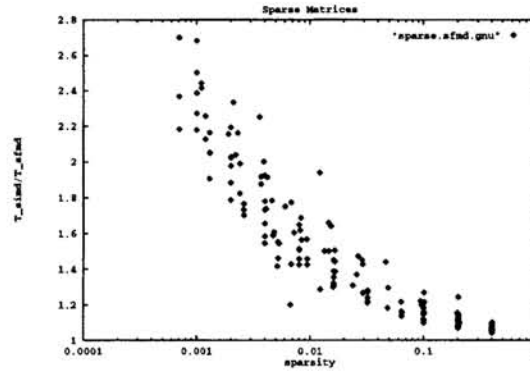

Figure 1: Sparse matrices: speedup vs. sparsity

numbers of data features, so distributing models and data to local memories is practical.

To examine the effect of SFMD on this form of model matching, we performed some simple simulations. To match a model with $D$ features to a set of $B$ data points, we attempt to match the first model feature with each data point in order, with some probability of success, $p_{match}$. If we succeed, we attempt to match the second model feature with one of the remaining $B - 1$ data points, and so on. If we match all $D$ features, we then check for global consistency of the correspondence, with some probability of success, $p_{check}$. This procedure is equivalent to DFS in a tree with branching factor $B - d$ at level $d$ of the tree, $1 \leq d \leq D$, where the probability of expanding any given node is $p_{match}$, and the probability of stopping the search at any given leaf is $1 - p_{check}$.

By writing the search as an iteration managing an explicit stack, one obtains a loop with some common code and some code conditional on whether the current node has any child nodes left to be expanded. The bulk of the "no-child" code deals with leaf nodes, consisting of testing for global consistency and recording solutions. The relative performance of SIMD and SFMD thus depends mainly on the probability, $p_{leaf}$, that the node being traversed is a leaf. If, for each iteration, the time for the leaf code is taken to be 1, that for common code is $t$, and that for the non-leaf code is $k$, then

$$T_{simd}/T_{sfmd} = \frac{t + k + 1}{t + (1 - p)k + p}. \tag{3}$$

Panel 1 of figure 2 shows values of $p$ from a variety of simulations of ITS, with $B, D \in \{8, 10, 12, 14, 16\}$, $p_{match} \in \{0.1, 0.2, 1/B\}$, $p_{check} \in \{0, 1\}$. Grimson (1990) reports searches on realistic data of around 5000-10000 expansions; this corresponds to $p \approx 0.2 - 0.4$. Panel 2 of figure 2 shows how equation 3 behaves for $p$ in this regime and for realistic values of $k$. We see speedups in the range 2–4 unless the leaf code is very small. In fact, the code for global consistency checking is typically larger than that for local consistency, corresponding to $\log_2 k < 0$.

## 8   OTHER USES

There are a number of uses for SFMD, other than model matching. First, common "subroutines" involving branching may be kept in the IM. Analysis of code for IEEE floating point emulation on an SIMD machine shows an expected 2x improvement by

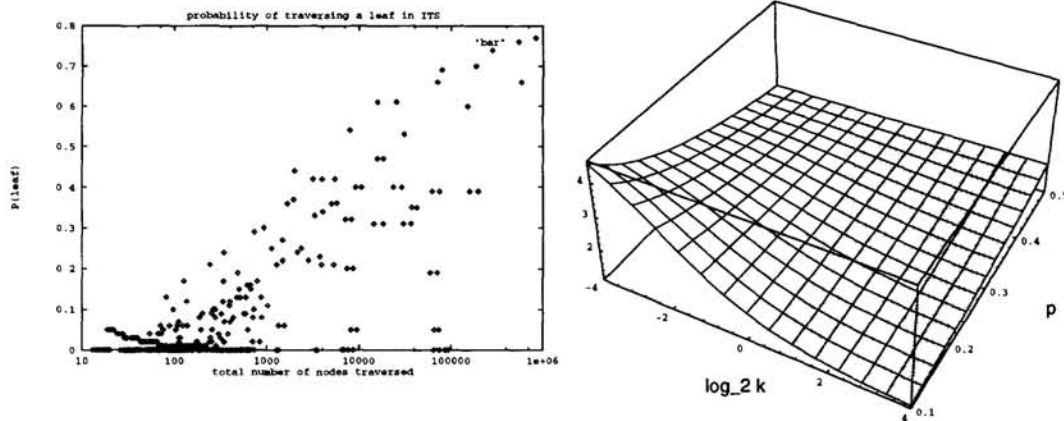

Figure 2: DFS speedup. Panel 1 shows the probability, $p$, of traversing a leaf. Panel 2 plots equation 3 for realistic values of $p$ and $k$, with $t = 0.1$.

using SFMD. Second, simple PE-local searches and sorts should show a significant, sub-2x, improvement in expected time. Third, more speculatively, different PEs can execute entirely different tasks by having the SFMD block consist of a single (nested) if-then-else. This would allow a form of (highly synchronized) pipeline parallelism by communicating results in SIMD mode after the end of the SFMD block.

## 9   CONCLUSION

We have introduced the SFMD computation model as a natural way of implementing the common task of model matching, and have shown how it extends SIMD computing, giving it greater flexibility and power. SFMD functionality can easily, and relatively cheaply, be added to existing SIMD designs already having a high degree of processor autonomy. The addition can be made without altering the user's programming model or environment. We have argued that technology trends will force multiple-processor-per-chip systems to increase processor complexity and memory, rather than increase the number of processors model per chip, and believe that the SFMD model is a natural step in that evolution.

### Acknowledgements

The first author gratefully acknowledges support under ARPA/ONR grants N00014-94-C-0130, N00014-92-J-4062, and N00014-94-1-0071.

## Footnotes

[1]E.g., due to limited pin count, pad area, and slower clock off-chip.

[2]For example, if the distributed code is a subroutine that will be encountered again.

[3]We assume the assignment of rows to PEs is independent of the number of nonzero elements in the rows. If not, then for $N \gg P$, simply sorting rows by number of elements and then assigning row $i$ to processor $i \bmod P$ is a good enough packing heuristic to make $T_{simd} \approx T_{sfmd}$.

[4]See (Grimson 1990) for a complete description of ITS and for the complexity results alluded to here.

### References

Grimson, W. E. L. (1990), *Object Recognition by Computer: The Role of Geometric Constraints*, MIT Press.

Hammerstrom, D. (1990), A VLSI architecture for high-performance, low-cost, on-chip learning, *in* 'The Proceedings of the IJCNN'.

Suetens, P., Fua, P. & Hanson, A. J. (1992), 'Computational strategies for object recognition', *Computing Surveys* **24**(1), 5 – 61.